# Randomized PCA Algorithms with Regret Bounds that are Logarithmic in the Dimension

**Manfred K. Warmuth**
Computer Science Department
University of California - Santa Cruz
manfred@cse.ucsc.edu

**Dima Kuzmin**
Computer Science Department
University of California - Santa Cruz
dima@cse.ucsc.edu

## Abstract

We design an on-line algorithm for Principal Component Analysis. In each trial the current instance is projected onto a probabilistically chosen low dimensional subspace. The total expected quadratic approximation error equals the total quadratic approximation error of the best subspace chosen in hindsight plus some additional term that grows linearly in dimension of the subspace but logarithmically in the dimension of the instances.

## 1   Introduction

In Principal Component Analysis the $n$-dimensional data instances are projected into a $k$-dimensional subspace ($k < n$) so that the total quadratic approximation error is minimized. After centering the data, the problem is equivalent to finding the eigenvectors of the $k$ largest eigenvalues of the data covariance matrix.

We develop a probabilistic on-line version of PCA: in each trial the algorithm chooses a $k$-dimensional projection matrix $\boldsymbol{P}^t$ based on some internal parameter; then an instance $\boldsymbol{x}^t$ is received and the algorithm incurs loss $\|\boldsymbol{x}^t - \boldsymbol{P}^t\boldsymbol{x}^t\|_2^2$; finally the internal parameter is updated. The goal is to obtain algorithms whose total loss in all trials is close to the smallest total loss of any $k$-dimensional subspace $\boldsymbol{P}$ chosen in hindsight.

We first develop our algorithms in the expert setting of on-line learning. The algorithm maintains a mixture vector over the $n$ experts. At the beginning of trial $t$ the algorithm chooses a subset $P^t$ of $k$ experts based on the current mixture vector $\boldsymbol{w}^t$. It then receives a loss vector $\boldsymbol{\lambda}^t \in [0..1]^n$ and incurs loss equal to the remaining $n - k$ components of the loss vector, i.e. $\sum_{i \in \{1,\ldots,n\}-P^t} \ell_i^t$. Finally it updates its mixture vector to $\boldsymbol{w}^{t+1}$. Note that now the subset $P^t$ corresponds to the subspace onto which we "project", i.e. we incur no loss on the $k$ components of $P^t$ and are charged only for the remaining $n - k$ components.

The trick is to maintain a mixture vector $\boldsymbol{w}^t$ as a parameter with the additional constraint that $\boldsymbol{w}_i^t \leq \frac{1}{n-k}$. We will show that these constrained mixture vectors represent an implicit mixture over subsets of experts of size $n - k$, and given $\boldsymbol{w}^t$ we can efficiently sample from the implicit mixture and use it to predict. This gives an on-line algorithm whose total loss is close to the smallest $n-k$ components of $\sum_t \boldsymbol{\lambda}^t$ and this algorithm generalizes to an on-line PCA algorithm when the mixture vectors are replaced by density matrices whose eigenvalues are bounded by $\frac{1}{n-k}$. Now the constrained density matrices represent implicit mixtures of the $(n - k)$-dimensional subspaces. The complementary $k$-dimensional space is used to project the current instance.

## 2 Standard PCA and On-line PCA

Given a sequence of data vectors $\boldsymbol{x}^1, \ldots, \boldsymbol{x}^T$, the goal is to find a low-dimensional approximation of this data that minimizes the 2-norm approximation error. Specifically, we want to find a rank $k$ projection matrix $\boldsymbol{P}$ and a bias vector $\boldsymbol{b} \in \mathbb{R}^n$ such that the following cost function is minimized:

$$\text{loss}(\boldsymbol{P}, \boldsymbol{b}) = \sum_{t=1}^{T} \|\boldsymbol{x}^t - (\boldsymbol{P}\boldsymbol{x}^t + \boldsymbol{b})\|_2^2.$$

Differentiating and solving for $\boldsymbol{b}$ gives us $\boldsymbol{b} = (\boldsymbol{I} - \boldsymbol{P})\,\bar{\boldsymbol{x}}$, where $\bar{\boldsymbol{x}}$ is the data mean. Substituting this bias $\boldsymbol{b}$ into the loss we obtain

$$\text{loss}(\boldsymbol{P}) = \sum_{t=1}^{T} \|(\boldsymbol{I} - \boldsymbol{P})(\boldsymbol{x}^t - \bar{\boldsymbol{x}})\|_2^2 = \sum_{t=1}^{T} (\boldsymbol{x}^t - \bar{\boldsymbol{x}})^{\top}(\boldsymbol{I} - \boldsymbol{P})^2(\boldsymbol{x}^t - \bar{\boldsymbol{x}}).$$

Since $\boldsymbol{I} - \boldsymbol{P}$ is a projection matrix, $(\boldsymbol{I} - \boldsymbol{P})^2 = \boldsymbol{I} - \boldsymbol{P}$, and we get:

$$\text{loss}(\boldsymbol{P}) = \text{tr}((\boldsymbol{I} - \boldsymbol{P}) \sum_{i=1}^{T} (\boldsymbol{x}_i - \bar{\boldsymbol{x}})(\boldsymbol{x}_i - \bar{\boldsymbol{x}})^{\top}) = \text{tr}(\underbrace{(\boldsymbol{I} - \boldsymbol{P})}_{\text{rank } n-k} \boldsymbol{C}) = \text{tr}(\boldsymbol{C}) - \text{tr}(\underbrace{\boldsymbol{P}}_{\text{rank } k} \boldsymbol{C}),$$

where $\boldsymbol{C}$ is the data covariance matrix. Therefore the loss is minimized over $(n - k)$-dimensional subspaces and this is equivalent to maximizing over $k$-dimensional subspaces.

In the on-line setting, learning proceeds in trials. (For the sake of simplicity we are not using a bias term at this point.) At trial $t$, the algorithm chooses a rank $k$ projection matrix $\boldsymbol{P}^t$. It then receives an instance $\boldsymbol{x}^t$ and incurs loss $\|\boldsymbol{x}^t - \boldsymbol{P}^t\boldsymbol{x}^t\|_2^2 = \text{tr}((\boldsymbol{I} - \boldsymbol{P}^t)\,\boldsymbol{x}^t(\boldsymbol{x}^t)^{\top})$. Our goal is to obtain an algorithm whose total loss over a sequence of trials $\sum_{t=1}^{T} \text{tr}((\boldsymbol{I} - \boldsymbol{P}^t)\,\boldsymbol{x}^t(\boldsymbol{x}^t)^{\top})$ is close to the total loss of the best rank $k$ projection matrix $\boldsymbol{P}$, i.e. $\inf_{\boldsymbol{P}} \text{tr}((\boldsymbol{I} - \boldsymbol{P}) \sum_{t=1}^{T} \boldsymbol{x}^t(\boldsymbol{x}^t)^{\top})$. Note that the latter loss is equal to the loss of standard PCA on data sequence $\boldsymbol{x}^1, \ldots, \boldsymbol{x}^T$ (assuming the data is centered).

## 3 Choosing a Subset of Experts

Recall that projection matrices are symmetric positive definite matrices with eigenvalues in $\{0, 1\}$. Thus a rank $k$ projection matrix can be written as $\boldsymbol{P} = \sum_{i=1}^{k} \boldsymbol{p}_i \boldsymbol{p}_i^{\top}$, where the $\boldsymbol{p}_i$ are the $k$ orthonormal vectors forming the basis of the subspace. Assume for the moment that the eigenvectors are restricted to be standard basis vectors. Now projection matrices become diagonal matrices with entries in $\{0, 1\}$, where the number of ones is the rank. Also, the trace of a product of such a diagonal projection matrix and any symmetric matrix becomes a dot product between the diagonals of both matrices and the whole problem reduces to working with vectors: the rank $k$ projection matrices reduce to vectors with $k$ ones and $n - k$ zeros and the diagonal of the symmetric matrix may be seen as a loss vector $\boldsymbol{\lambda}^t$. Our goal now is to develop on-line algorithms for finding the lowest $n - k$ components of the loss vectors $\boldsymbol{\lambda}^t$ so that the total loss is close the to the lowest $n - k$ components of $\sum_{t=1}^{T} \boldsymbol{\lambda}^t$. Equivalently, we want to find the highest $k$ components in $\boldsymbol{\lambda}^t$.

We begin by developing some methods for dealing with subsets of components. For convenience we encode such subsets as probability vectors: we call $\boldsymbol{r} \in [0, 1]^n$ an $m$-*corner* if it has $m$ components set to $\frac{1}{m}$ and the remaining $n - m$ components set to zero. At trial $t$ the algorithm chooses an $(n - k)$-corner $\boldsymbol{r}^t$. It then receives a loss vector $\boldsymbol{\lambda}^t$ and incurs loss $(n - k)\,\boldsymbol{r}^t \cdot \boldsymbol{\lambda}^t$.

Let $A_m^n$ consist of all convex combinations of $m$-corners. In other words, $A_m^n$ is the convex hull of the $\binom{n}{m}$ $m$-corners. Clearly any component $w_i$ of a vector $\boldsymbol{w}$ in $A_m^n$ is at most $\frac{1}{m}$ because it is a convex combination of numbers in $[0..\frac{1}{m}]$. Therefore $A_m^n \subseteq B_m^n$, where $B_m^n$ is the set of $n$-dimensional vectors $\boldsymbol{w}$ for which $|\boldsymbol{w}| = \sum_i w_i = 1$ and $0 \leq w_i \leq \frac{1}{m}$, for all $i$. The following theorem implies that $A_m^n = B_m^n$:

**Theorem 1.** *Algorithm 1 produces a convex combination[1] of at most $n$ $m$-corners for any vector in* $B_m^n$.

---

**Algorithm 1** Mixture Construction

---

    **input** $1 \leq m < n$ and $\boldsymbol{w} \in B_m^n$
    **repeat**
        Let $\boldsymbol{r}$ be a corner whose $m$ components correspond to nonzero components of $\boldsymbol{w}$
            and contain all the components of $\boldsymbol{w}$ that are equal to $\frac{|\boldsymbol{w}|}{m}$
        Let $s$ be the smallest of the $m$ chosen components in $\boldsymbol{w}$
            and $l$ be the largest value of the remaining $n - m$ components
$$\boldsymbol{w} := \boldsymbol{w} - \overbrace{\min(m\,s, |w| - m\,l)}^{p} \, \boldsymbol{r} \text{ and } \textbf{output } p\,\boldsymbol{r}$$
    **until** $\boldsymbol{w} = \boldsymbol{0}$

---

*Proof.* Let $b(\boldsymbol{w})$ be the number of *boundary* components in $\boldsymbol{w}$, i.e. $b(\boldsymbol{w}) := |\{i : w_i \text{ is } 0 \text{ or } \frac{|\boldsymbol{w}|}{m}\}|$. Let $\widetilde{B}_m^n$ be all vectors $\boldsymbol{w}$ such that $0 \leq w_i \leq \frac{|\boldsymbol{w}|}{m}$, for all $i$. If $b(\boldsymbol{w}) = n$, then $\boldsymbol{w}$ is either a corner or $\boldsymbol{0}$. The loop stops when $\boldsymbol{w} = \boldsymbol{0}$. If $\boldsymbol{w}$ is a corner then it takes one iteration to arrive at $\boldsymbol{0}$. We show if $\boldsymbol{w} \in \widetilde{B}_m^n$ and $\boldsymbol{w}$ is neither a corner nor $\boldsymbol{0}$, then the successor $\widehat{\boldsymbol{w}} \in \widetilde{B}_m^n$ and $b(\widehat{\boldsymbol{w}}) > b(\boldsymbol{w})$. Clearly, $\widehat{\boldsymbol{w}} \geq \boldsymbol{0}$, because the amount that is subtracted in the $m$ components of the corner is at most as large as the corresponding components of $\boldsymbol{w}$. We next show that $\widehat{w}_i \leq \frac{|\widehat{\boldsymbol{w}}|}{m}$. If $i$ belongs to the corner then $\widehat{w}_i = w_i - \frac{p}{m} \leq \frac{|\boldsymbol{w}| - p}{m} = \frac{|\widehat{\boldsymbol{w}}|}{m}$. Otherwise $\widehat{w}_i = w_i \leq l$, and $l \leq \frac{|\widehat{\boldsymbol{w}}|}{m}$ follows from the fact that $p \leq |\boldsymbol{w}| - m\,l$. This proves that $\widehat{\boldsymbol{w}} \in \widetilde{B}_m^n$.

For showing that $b(\widehat{\boldsymbol{w}}) > b(\boldsymbol{w})$ first observe that all boundary components in $\boldsymbol{w}$ remain boundary components in $\widehat{\boldsymbol{w}}$: zeros stay zeros and if $w_i = \frac{|\boldsymbol{w}|}{m}$ then $i$ is included in the corner and $\widehat{w}_i = \frac{|\boldsymbol{w}| - p}{m} = \frac{|\widehat{\boldsymbol{w}}|}{m}$. However, the number of boundary components is increased at least by one because the components corresponding to $s$ and $l$ are both non-boundary components in $\boldsymbol{w}$ and at least one of them becomes a boundary point in $\widehat{\boldsymbol{w}}$: if $p = m\,s$ then the component corresponding to $s$ in $\boldsymbol{w}$ is $s - \frac{p}{m} = 0$ in $\widehat{\boldsymbol{w}}$ and if $p = |\boldsymbol{w}| - m\,l$ then the component corresponding to $l$ in $\boldsymbol{w}$ is $l = \frac{|\boldsymbol{w}| - p}{m} = \frac{|\widehat{\boldsymbol{w}}|}{m}$. It follows that it may take up to $n$ iterations to arrive at a corner which has $n$ boundary components and one more iteration to arrive at $\boldsymbol{0}$. Finally note that there is no weight vector $\boldsymbol{w} \in \widetilde{B}_m^n$ s.t. $b(\boldsymbol{w}) = n - 1$ and therefore the size of the produced linear combination is at most $n$. More precisely, the size is at most $n - b(\boldsymbol{w})$ if $n - b(\boldsymbol{w}) \leq n - 2$ and one if $\boldsymbol{w}$ is a corner.

The algorithm produces a linear combinations of corners, i.e. $\boldsymbol{w} = \sum_j p_j \boldsymbol{r}_j$. Since $p_j \geq 0$ and all $|\boldsymbol{r}_j| = 1$, $\sum_j p_j = 1$ and we actually have a convex combination.   □

**Fact 1.** *For any loss vector $\boldsymbol{\lambda}$, the following corner has the smallest loss of any convex combination of corners in $A_m^n = B_m^n$: Greedily pick the component of minimum loss ($m$ times).*

How can we use the above construction and fact? It seems too hard to maintain information about all $\binom{n}{n-k}$ corners of size $n-k$. However, the best corner is also the best convex combination of corners, i.e. the best from the set $A_{n-k}^n$ where each member of this set is given by $\binom{n}{n-k}$ coefficients. Luckily, this set of convex combinations equals $B_{n-k}^n$ and it takes $n$ coefficients to specify a member in that set. Therefore we can search for the best hypothesis in the set $B_{n-k}^n$ and for any such hypothesis we can always construct a convex combination (of size $\leq n$) of $(n-k)$-corners which has the same expected loss for each loss vector. This means that any algorithm predicting with a hypothesis vector in $B_{n-k}^n$ can be converted to an algorithm that probabilistically chooses an $(n-k)$-corner. Finally, the set $P^t$ of the $k$ components missed by the chosen $(n-k)$-corner corresponds to the subspace we project onto.

Algorithm 2 spells out the details for this approach. The algorithm chooses a corner probabilistically and $(n-k)\,\boldsymbol{w}^t \cdot \boldsymbol{\lambda}^t$ is the expected loss in one trial. The projection $\widehat{\boldsymbol{w}}^t$ onto $B_{n-k}^n$ can be achieved as follows: find the smallest $l$ s.t. capping the largest $l$ components to $\frac{1}{n-k}$ and rescaling the remaining $n-l$ weights to total weight $1 - \frac{l}{n-k}$ makes none of the rescaled weights go above $\frac{1}{n-k}$. The simplest

algorithm starts with sorting the weights and then searches for $l$ with a binary search. However, a linear algorithm that recursively uses the median is given in [HW01].

---

**Algorithm 2** Capped Weighted Majority Algorithm

---

**input**: $1 \le k < n$ and an initial probability vector $\boldsymbol{w}^1 \in B_{n-k}^n$
**for** $t = 1$ to $T$ **do**
    Decompose $\boldsymbol{w}^t$ as $\sum_j p_j \boldsymbol{r}_j$ with Algorithm 1, where $m = n - k$
    Draw a corner $\boldsymbol{r} = \boldsymbol{r}_j$ with probability $p_j$
    Let $P^t$ be the $k$ components outside the drawn corner
    Receive loss vector $\boldsymbol{\lambda}^t$
    Incur loss $(n-k)\,\boldsymbol{r} \cdot \boldsymbol{\lambda}^t = \sum_{i \in \{1,\dots,n\} - P^t} \ell_i^t$.
    $\widehat{w}_i^t := w_i^t \exp(-\eta \ell_i^t) / Z$, where $Z$ normalizes the weights to one
    $\boldsymbol{w}^{t+1} := \operatorname*{argmin}_{\boldsymbol{w} \in B_{n-k}^n} d(\boldsymbol{w}, \widehat{\boldsymbol{w}}^t)$
**end for**

---

When $k = n - 1$, $n - k = 1$ and $B_1^n$ is the entire probability simplex. In this case the call to Algorithm 1 and the projection onto $B_1^n$ are vacuous and we get the standard Randomized Weighted Majority algorithm [LW94][2] with loss vector $\boldsymbol{\lambda}^t$.

Let $d(\boldsymbol{u}, \boldsymbol{w})$ denote the relative entropy between two probability vectors: $d(\boldsymbol{u}, \boldsymbol{w}) = \sum_i u_i \log \frac{u_i}{w_i}$.

**Theorem 2.** *On an arbitrary sequence of loss vectors $\boldsymbol{\lambda}^1, \dots, \boldsymbol{\lambda}^T \in [0,1]^n$, the total expected loss of Algorithm 2 is bounded as follows:*

$$(n-k) \sum_{t=1}^T \boldsymbol{w}^t \cdot \boldsymbol{\lambda}^t \le (n-k) \frac{\eta \sum_{t=1}^T \boldsymbol{u} \cdot \boldsymbol{\lambda}^t + d(\boldsymbol{u}, \boldsymbol{w}^1) - d(\boldsymbol{u}, \boldsymbol{w}^{T+1})}{1 - \exp(-\eta)},$$

*for any learning rate $\eta > 0$ and comparison vector $\boldsymbol{u} \in B_{n-k}^n$.*

*Proof.* The update for $\widehat{\boldsymbol{w}}^t$ in Algorithm 2 is the update of the Continuous Weighted Majority for which the following basic inequality is known (essentially [LW94], Lemma 5.3):

$$d(\boldsymbol{u}, \boldsymbol{w}^t) - d(\boldsymbol{u}, \widehat{\boldsymbol{w}}^t) \ge -\eta\, \boldsymbol{u} \cdot \boldsymbol{\lambda}^t + \boldsymbol{w}^t \cdot \boldsymbol{\lambda}^t (1 - \exp(-\eta)). \tag{1}$$

The weight vector $\boldsymbol{w}^{t+1}$ is a Bregman projection of vector $\widehat{\boldsymbol{w}}^t$ onto the convex set $B_{n-k}^n$. For such projections the Generalized Pythagorean Theorem holds (see e.g [HW01] for details):

$$d(\boldsymbol{u}, \widehat{\boldsymbol{w}}^t) \ge d(\boldsymbol{u}, \boldsymbol{w}^{t+1}) + d(\boldsymbol{w}^{t+1}, \widehat{\boldsymbol{w}}^t)$$

Since Bregman divergences are non-negative, we can drop the $d(\boldsymbol{w}^{t+1}, \widehat{\boldsymbol{w}}^t)$ term and get the following inequality:

$$d(\boldsymbol{u}, \widehat{\boldsymbol{w}}^t) - d(\boldsymbol{u}, \boldsymbol{w}^{t+1}) \ge 0, \text{ for } \boldsymbol{u} \in B_{n-k}^n.$$

Adding this to the previous inequality we get:

$$d(\boldsymbol{u}, \boldsymbol{w}^t) - d(\boldsymbol{u}, \boldsymbol{w}^{t+1}) \ge -\eta\, \boldsymbol{u} \cdot \boldsymbol{\lambda}^t + \boldsymbol{w}^t \cdot \boldsymbol{\lambda}^t (1 - \exp(-\eta))$$

By summing over $t$, multiplying by $n - k$, and dividing by $1 - \exp(-\eta)$, the bound follows. $\qquad\square$

## 4 On-line PCA

In this context *(matrix) corners* are density matrices with $m$ eigenvalues equal to $\frac{1}{m}$ and the rest are 0. Also the set $\boldsymbol{\mathcal{A}}_m^n$ consists of all convex combinations of such corners. The maximum eigenvalue of a convex combination of symmetric matrices is at most as large as the maximum eigenvalue of any of the matrices ([Bha97], Corollary III.2.2). Therefore each convex combination of corners is

a density matrix whose eigenvalues are bounded by $\frac{1}{m}$ and $\mathcal{A}_m^n \subseteq \mathcal{B}_m^n$, where $\mathcal{B}_m^n$ consists of all density matrices whose maximum eigenvalue is at most $\frac{1}{m}$. Assume we have some density matrix $\boldsymbol{W} \in \mathcal{B}_m^n$ with eigendecomposition $\boldsymbol{\mathcal{W}} \operatorname{diag}(\boldsymbol{\omega}) \boldsymbol{\mathcal{W}}^\top$. Algorithm 1 can be applied to the vector of eigenvalues $\boldsymbol{\omega}$ of this density matrix. The output convex combination of up to $n$ diagonal corners $\boldsymbol{\omega} = \sum_j p_j \boldsymbol{r}_j$ can be turned into a convex combination of matrix corners that expresses the density matrix: $\boldsymbol{W} = \sum_j p_j \boldsymbol{\mathcal{W}} \operatorname{diag}(\boldsymbol{r}_j) \boldsymbol{\mathcal{W}}^\top$. It follows that $\mathcal{A}_m^n = \mathcal{B}_m^n$ as in the diagonal case.

**Theorem 3.** *For any symmetric matrix $\boldsymbol{S}$, $\min_{\boldsymbol{W} \in \mathcal{B}_m^n} \operatorname{tr}(\boldsymbol{W}\boldsymbol{S})$ attains its minimum at the following matrix corner: greedily choose orthogonal eigenvectors of $\boldsymbol{S}$ of minimum eigenvalue ($m$ times).*

*Proof.* Let $\lambda^\downarrow(\boldsymbol{W})$ denote the vector of eigenvalues of $\boldsymbol{W}$ in descending order and let $\lambda^\uparrow(\boldsymbol{S})$ be the same vector of $\boldsymbol{S}$ but in ascending order. Since both matrices are symmetric, $\operatorname{tr}(\boldsymbol{W}\boldsymbol{S}) \geq \lambda^\downarrow(\boldsymbol{W}) \cdot \lambda^\uparrow(\boldsymbol{S})$ ([MO79], Fact H.1.h of Chapter 9). Since $\lambda^\downarrow(\boldsymbol{W}) \in B_m^n$, the dot product is minimized and the inequality is tight when $\boldsymbol{W}$ is an $m$-corner corresponding to the $m$ smallest eigenvalues of $\boldsymbol{S}$. Also the greedy algorithm finds the solution (see Fact 1 of this paper). $\qquad\square$

Algorithm 2 generalizes to the matrix setting. The Weighted Majority update is replaced by the corresponding matrix version which employs the matrix exponential and matrix logarithm [WK06] (The update can be seen as a special case of the Matrix Exponentiated Gradient update [TRW05]).

The following theorem shows that for the projection we can keep the eigensystem fixed. Here $\Delta(\boldsymbol{U}, \boldsymbol{W})$ denotes the quantum relative entropy $\operatorname{tr}(\boldsymbol{U}(\log \boldsymbol{U} - \log \boldsymbol{W}))$.

**Theorem 4.** *Projecting a density matrix onto $\mathcal{B}_m^n$ w.r.t. the quantum relative entropy is equivalent to projecting the vector of eigenvalues w.r.t. the "normal" relative entropy: If $\boldsymbol{W}$ has the eigendecomposition $\boldsymbol{\mathcal{W}} \operatorname{diag}(\boldsymbol{\omega}) \boldsymbol{\mathcal{W}}^\top$, then*

$$\operatorname*{argmin}_{\boldsymbol{U} \in \mathcal{B}_m^n} \Delta(\boldsymbol{U}, \boldsymbol{W}) = \boldsymbol{\mathcal{W}} \boldsymbol{u}^* \boldsymbol{\mathcal{W}}^\top, \text{ where } \boldsymbol{u}^* = \operatorname*{argmin}_{\boldsymbol{u} \in B_m^n} d(\boldsymbol{u}, \boldsymbol{\omega}).$$

*Proof.* If $\lambda^\downarrow(\boldsymbol{S})$ denotes the vector of eigenvalues of a symmetric matrix $\boldsymbol{S}$ arranged in descending order, then $\operatorname{tr}(\boldsymbol{S}\boldsymbol{T}) \leq \lambda^\downarrow(\boldsymbol{S}) \cdot \lambda^\downarrow(\boldsymbol{T})$ ([MO79], Fact H.1.g of Chapter 9). This implies that $\operatorname{tr}(\boldsymbol{U} \log \boldsymbol{W}) \leq \lambda^\downarrow(\boldsymbol{U}) \cdot \log \lambda^\downarrow(\boldsymbol{W})$ and $\Delta(\boldsymbol{U}, \boldsymbol{W}) \geq d(\lambda^\downarrow(\boldsymbol{U}), \lambda^\downarrow(\boldsymbol{W}))$. Therefore $\min_{\boldsymbol{U} \in \mathcal{B}_m^n} \Delta(\boldsymbol{U}, \boldsymbol{W}) \geq \min_{\boldsymbol{u} \in B_m^n} d(\boldsymbol{u}, \boldsymbol{\omega})$ and if $\boldsymbol{u}^*$ minimizes the r.h.s. then $\boldsymbol{\mathcal{W}} \operatorname{diag}(\boldsymbol{u}^*) \boldsymbol{\mathcal{W}}^\top$ minimizes the l.h.s. because $\Delta(\boldsymbol{\mathcal{W}} \operatorname{diag}(\boldsymbol{u}^*) \boldsymbol{\mathcal{W}}, \boldsymbol{W}) = d(\boldsymbol{u}^*, \boldsymbol{\omega})$. $\qquad\square$

---

**Algorithm 3** On-line PCA algorithm

---

 **input**: $1 \leq k < n$ and an initial density matrix $\boldsymbol{W}^1 \in \mathcal{B}_{n-k}^n$
 **for** $t = 1$ to $T$ **do**
  Perform eigendecomposition $\boldsymbol{W}^t = \boldsymbol{\mathcal{W}} \boldsymbol{\omega} \boldsymbol{\mathcal{W}}^\top$
  Decompose $\boldsymbol{\omega}$ as $\sum_j p_j \boldsymbol{r}_j$ with Algorithm 1, where $m = n - k$
  Draw a corner $\boldsymbol{r} = \boldsymbol{r}_j$ with probability $p_j$
  Form a matrix corner $\boldsymbol{R} = \boldsymbol{\mathcal{W}} \operatorname{diag}(\boldsymbol{r}) \boldsymbol{\mathcal{W}}^\top$
  Form a rank $k$ projection matrix $\boldsymbol{P}^t = \boldsymbol{I} - (n - k)\boldsymbol{R}$
  Receive data instance vector $\boldsymbol{x}^t$
  Incur loss $\|\boldsymbol{x}^t - \boldsymbol{P}^t \boldsymbol{x}^t\|_2^2 = \operatorname{tr}((\boldsymbol{I} - \boldsymbol{P}^t)\boldsymbol{x}^t(\boldsymbol{x}^t)^\top)$
  $\widehat{\boldsymbol{W}}^t = \exp(\log \boldsymbol{W}^t - \eta\, \boldsymbol{x}^t(\boldsymbol{x}^t)^\top) / Z$, where $Z$ normalizes the trace to 1
  $\boldsymbol{W}^{t+1} := \operatorname*{argmin}_{\boldsymbol{W} \in \mathcal{B}_{n-k}^n} \Delta(\boldsymbol{W}, \widehat{\boldsymbol{W}}^t)$
 **end for**

---

The expected loss in trial $t$ of this algorithm is given by $(n - k)\operatorname{tr}(\boldsymbol{W}^t \boldsymbol{x}^t(\boldsymbol{x}^t)^\top)$

**Theorem 5.** *For an arbitrary sequence of data instances $\boldsymbol{x}^1, \dots, \boldsymbol{x}^T$ of 2-norm at most one, the total expected loss of the algorithm is bounded as follows:*

$$\sum_{t=1}^{T} (n - k)\operatorname{tr}(\boldsymbol{W}^t \boldsymbol{x}^t(\boldsymbol{x}^t)^\top) \leq (n - k)\frac{\eta \sum_{t=1}^{T} \operatorname{tr}(\boldsymbol{U}\boldsymbol{x}^t(\boldsymbol{x}^t)^\top) + \Delta(\boldsymbol{U}, \boldsymbol{W}^1) - \Delta(\boldsymbol{U}, \boldsymbol{W}^T)}{1 - \exp(-\eta)},$$

*for any learning rate $\eta > 0$ and comparator density matrix $\boldsymbol{U} \in \mathcal{B}_{n-k}^n$.*[3]

*Proof.* The update for $\widehat{\boldsymbol{W}}^t$ is a density matrix version of the standard Weighted Majority update which was used for variance minimization along a single direction (i.e. $k = n - 1$) in [WK06]. The basic inequality (1) for that update becomes:

$$\Delta(\boldsymbol{U}, \boldsymbol{W}^t) - \Delta(\boldsymbol{U}, \widehat{\boldsymbol{W}}^t) \geq -\eta \operatorname{tr}(\boldsymbol{U}\boldsymbol{x}^t(\boldsymbol{x}^t)^\top) + \operatorname{tr}(\boldsymbol{W}^t\boldsymbol{x}^t(\boldsymbol{x}^t)^\top)(1 - \exp(-\eta))$$

As in the proof of Theorem 2 of this paper, the Generalized Pythagorean theorem applies and dropping one term we get the following inequality:

$$\Delta(\boldsymbol{U}, \widehat{\boldsymbol{W}}^t) - \Delta(\boldsymbol{U}, \boldsymbol{W}^{t+1}) \geq 0, \text{ for } \boldsymbol{U} \in \mathcal{B}_{n-k}^n.$$

Adding this to the previous inequality we get:

$$\Delta(\boldsymbol{U}, \boldsymbol{W}^t) - \Delta(\boldsymbol{U}, \boldsymbol{W}^{t+1}) \geq -\eta \operatorname{tr}(\boldsymbol{U}\boldsymbol{x}^t(\boldsymbol{x}^t)^\top) + \operatorname{tr}(\boldsymbol{W}^t\boldsymbol{x}^t(\boldsymbol{x}^t)^\top)(1 - \exp(-\eta))$$

By summing over $t$, multiplying by $n - k$, and dividing by $1 - \exp(-\eta)$, the bound follows. $\qquad\square$

It is easy to see that $\Delta(\boldsymbol{U}, \boldsymbol{W}^1) \leq (n - k) \log \frac{n}{n-k}$. If $k \leq n/2$, then this is further bounded by $k \log \frac{n}{k}$. Thus, the r.h.s. is essentially linear in $k$, but logarithmic in the dimension $n$.

By tuning $\eta$ [CBFH$^+$97, FS97], we can get regret bounds of the form:

$$\text{(expected total loss of alg.) - (total loss best } k\text{-space)}$$
$$= \quad O\left(\sqrt{\text{(total loss of best } k\text{-subspace)} \, k \log \frac{n}{k}} + k \log \frac{n}{k}\right). \tag{2}$$

Using standard but significantly simplified conversion techniques from [CBFH$^+$97] based on the leave-one-out loss we also obtain algorithms with good regret bounds in the following model: the algorithm is given $T - 1$ instances drawn from a fixed but unknown distribution and produces a $k$-space based on those instances; it then receives a new instance from the same distribution. We can bound the expected loss on the last instance:

$$\text{(expected loss of alg.) - (expected loss best } k\text{-space)}$$
$$= \quad O\left(\sqrt{\frac{\text{(expected loss of best } k\text{-subspace)} \, k \log \frac{n}{k}}{T}} + \frac{k \log \frac{n}{k}}{T}\right). \tag{3}$$

## 5   Lower Bound

The simplest competitor to our on-line PCA algorithm is the algorithm that does standard (uncentered) PCA on all the data points seen so far. In the expert setting this algorithm corresponds to "projecting" to the $n - k$ experts that have minimum loss so far (where ties are broken arbitrarily). When $k = n - 1$, this becomes the *follow the leader* algorithm. It is easy to construct an adversary strategy for this type of deterministic algorithm (any $k$) that forces the on-line algorithm to incur $n$ times as much loss as the off-line algorithm. In contrast our algorithm is guaranteed to have expected additional loss (regret) of the order of square root of $k \ln n$ times the total loss of the best off-line algorithm. When the instances are diagonal matrices then our algorithm specializes to the standard expert setting and in that setting there are probabilistic lower bounds that show that our tuned bounds (2,3) are tight [CBFH$^+$97].

## 6   Simple Experiments

The above lower bounds do not justify our complicated algorithms for on-line PCA because natural data might be more benign. However natural data often shifts and we constructed a simple dataset of this type in Figure 1. The first 333 20-dimensional points were drawn from a Gaussian distribution with a rank 2 covariance matrix. This is repeated twice for different covariance matrices of rank

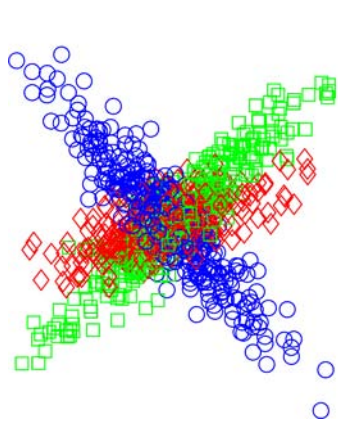

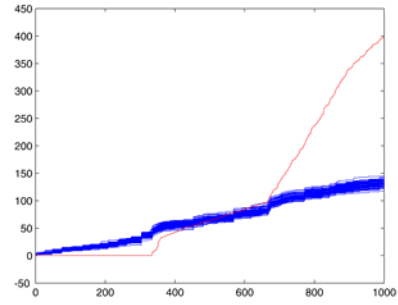

Figure 1: The data set used for the experiments. Different colors/symbols denote the data points that came from three different Gaussians with rank 2 covariance matrices. The data vectors are 20-dimensional but we plot only the first 3 dimensions.

Figure 2: The blue curve plots the total loss of on-line algorithm up to trial $t$ for 50 different runs (with $k = 2$ and $\eta$ fixed to one). Note that the variance of the losses is small. The red single curve plots the total loss of the best subspace of dimension 2 for the first $t$ points.

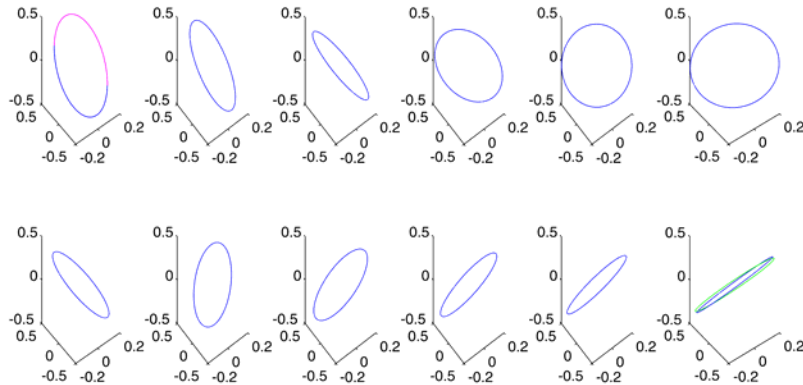

Figure 3: Behavior of the algorithm around a transition point between two distributions. Each ellipse depicts the projection matrix with the largest coefficient in the decomposition of $\boldsymbol{W}^t$. The transition sequence starts with the algorithm focused on the projection matrix for the first subset of data and ends with essentially the optimal matrix for the second subset. The depicted transition takes about 60 trials.

2. We compare the total loss of our on-line algorithm with the total loss of the best subspace for the first $t$ data points. During the first 333 datapoints the latter loss is zero since the first dataset is 2-dimensional, but after the third dataset is completed, the loss of any fixed off-line comparator is large. Figure 3 depicts how our algorithm transitions between datasets and exploits the on-lineness of the data. Randomly permuting the dataset removes the on-lineness and results in a plot where the total loss of the algorithm is somewhat above that of the off-line comparator (not shown).

Any simple "windowing algorithm" would also be able to detect the switches. Such algorithms are often unwieldy and we don't know any strong regret bounds for them. In the expert setting there is however a long line of research on shifting (see e.g. [BW02, HW98]). An algorithm that mixes a little bit of the uniform distribution into the current mixture vector is able to restart when the data switches. More importantly, an algorithm that mixes in a little bit of the past average density matrix is able to switch quickly to previously seen subspaces and to our knowledge windowing techniques cannot exploit this type of switching. Preliminary experiments on face image data indicate that the algorithms that accommodate switching work as expected, but more comprehensive experiments still need to be done.

# 7 Conclusions

We developed a new set of techniques for low dimensional approximation with provable bounds. Following [TRW05, WK06], we essentially lifted the algorithms and bounds developed for diagonal case to the matrix case. Are there general reductions?

The on-line PCA problem was also addressed in [Cra06]. However, that paper does not fully capture the PCA problem because their algorithm predicts with a full-rank matrix in each trial, whereas we predict with a probabilistically chosen projection matrix of the desired rank $k$. Furthermore, that paper proves bounds on the filtering loss, which are typically easier to prove, and it is not clear how this loss relates to the more standard regret bounds proven in this paper.

For the expert setting there are alternate techniques for designing on-line algorithms that do as well as the best subset of $n - k$ experts: set $\{i_1, \ldots, i_{n-k}\}$ receives weight proportional to $\exp(-\sum_j \ell_{i_j}^{<t}) = \prod_j \exp(-\ell_{i_j}^{<t})$. In this case we can get away with keeping only one weight per expert (the $i$th expert gets weight $\exp(-\ell_i^{<t})$) and then use dynamic programming to sum over sets (see e.g. [TW03] for this type of methods). With some more work, dynamic programming can also be applied for PCA. However, our new trick of using additional constraints on the eigenvalues is an alternative that avoids dynamic programming.

Many technical problems remain. For example we would like to enhance our algorithms to learn a bias as well and apply our low-dimensional approximation techniques to regression problems.

**Acknowledgment:** Thanks to Allen Van Gelder for valuable discussions re. Algorithm 1.

## Footnotes

[1]The existence of a convex combination of at most $n$ corners is implied by Carathéodory's theorem [Roc70], but the algorithm gives an effective construction.

[2]The original Weighted Majority algorithms were described for the absolute loss. The idea of using loss vectors instead was introduced in [FS97].

[3]The $\boldsymbol{x}^t(\boldsymbol{x}^t)^\top$ can replaced by symmetric matrices $\boldsymbol{S}^t$ whose eigenvalues have range at most one.

## References

[Bha97]    R. Bhatia. *Matrix Analysis*. Springer, Berlin, 1997.

[BW02]     Olivier Bousquet and Manfred K. Warmuth. Tracking a small set of experts by mixing past posteriors. *Journal of Machine Learning Research*, 3:363–396, 2002.

[CBFH+97]  N. Cesa-Bianchi, Y. Freund, D. Haussler, D. P. Helmbold, R. E. Schapire, and M. K. Warmuth. How to use expert advice. 44(3):427–485, 1997.

[Cra06]    Koby Crammer. Online tracking of linear subspaces. In *Proceedings of the 19th Annual Conference on Learning Theory (COLT 06)*, Pittsburg, June 2006. Springer.

[FS97]     Yoav Freund and Robert E. Schapire. A decision-theoretic generalization of on-line learning and an application to boosting. *Journal of Computer and System Sciences*, 55(1):119–139, August 1997.

[HW98]     Mark Herbster and Manfred Warmuth. Tracking the best expert. *Machine Learning*, 32(2):151–178, 1998. Earlier version in 12th ICML, 1995.

[HW01]     Mark Herbster and Manfred K. Warmuth. Tracking the best linear predictor. *Journal of Machine Learning Research*, 1:281–309, 2001.

[LW94]     N. Littlestone and M. K. Warmuth. The weighted majority algorithm. *Inform. Comput.*, 108(2):212–261, 1994.

[MO79]     A. W. Marshall and I. Olkin. *Inequalities: Theory of Majorization and its Applications*. Academic Press, 1979.

[Roc70]    R. Rockafellar. *Convex Analysis*. Princeton University Press, 1970.

[TRW05]    K. Tsuda, G. Rätsch, and M. K. Warmuth. Matrix exponentiated gradient updates for on-line learning and Bregman projections. *Journal of Machine Learning Research*, 6:995–1018, June 2005.

[TW03]     Eiji Takimoto and Manfred K. Warmuth. Path kernels and multiplicative updates. *Journal of Machine Learning Research*, 4:773–818, 2003.

[WK06]     Manfred K. Warmuth and Dima Kuzmin. Online variance minimization. In *Proceedings of the 19th Annual Conference on Learning Theory (COLT 06)*, Pittsburg, June 2006. Springer.
